# Subordinate class recognition using relational object models

**Aharon Bar Hillel**
Department of Computer Science
The Hebrew university of Jerusalem
aharonbh@cs.huji.ac.il

**Daphna Weinshall**
Department of Computer Science
The Hebrew university of Jerusalem
daphna@cs.huji.ac.il

## Abstract

We address the problem of sub-ordinate class recognition, like the distinction between different types of motorcycles. Our approach is motivated by observations from cognitive psychology, which identify parts as the defining component of basic level categories (like motorcycles), while sub-ordinate categories are more often defined by part properties (like 'jagged wheels'). Accordingly, we suggest a two-stage algorithm: First, a relational part based object model is learnt using unsegmented object images from the inclusive class (e.g., motorcycles in general). The model is then used to build a class-specific vector representation for images, where each entry corresponds to a model's part. In the second stage we train a standard discriminative classifier to classify subclass instances (e.g., cross motorcycles) based on the class-specific vector representation. We describe extensive experimental results with several subclasses. The proposed algorithm typically gives better results than a competing one-step algorithm, or a two stage algorithm where classification is based on a model of the sub-ordinate class.

## 1   Introduction

Human categorization is fundamentally hierarchical, where categories are organized in tree-like hierarchies. In this organization, higher nodes close to the root describe inclusive classes (like vehicles), intermediate nodes describe more specific categories (like motorcycles), and lower nodes close to the leaves capture fine distinctions between objects (e.g., cross vs. sport motorcycles). Intuitively one could expect such hierarchy to be learnt either bottom-up or top-down (or both), but surprisingly, this is *not* the case. In fact, there is a well defined intermediate level in the hierarchy, called *basic level*, which is learnt first [11]. In addition to learning, this level is more primary than both more specific and more inclusive levels, in terms of many other psychological, anthropological and linguistic measures.

The primary role of basic level categories seems related to the structure of objects in the world. In [13], Tversky & Hemenway promote the hypothesis that the explanation lies in the notion of parts. Their experiments show that basic level categories (like cars and flowers) are often described as a combination of distinctive parts (e.g., stem and petals), which are mostly unique. Higher (super-ordinate and more inclusive) levels are more often described by their function (e.g., 'used for transportation'), while lower (sub-ordinate and more specific) levels are often described by part properties (e.g., red petals) and other fine details. These points are illustrated in Fig. 1.

This computational characterization of human categorization finds parallels in computer vision and machine learning. Specifically, traditional work in pattern recognition focused on discriminating vectors of features, where the features are shared by all objects, with different values. If we make the analogy between features and parts, this level of analysis is appropriate for sub-ordinate categories. In this level different objects share parts but differ in the parts' values (e.g., red petals vs. yellow petals); this is called 'modified parts' in [13].

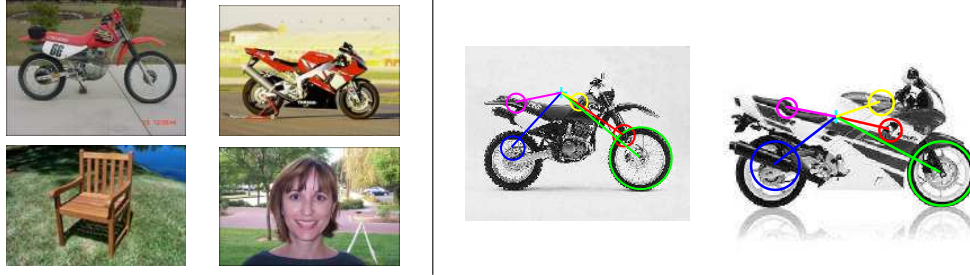

Figure 1: **Left** Examples of sub-ordinate and basic level classification. *Top row:* Two motorcycle subordinate classes, sport (right) and cross (left). As members of the same basic level category, they share the same part structure. *Bottom row:* Objects from different basic level categories, like a chair and a face, lack such natural part correspondence. **Right**. Several parts from a learnt motorcycle model as detected in cross and sport motorcycle images. Based on the part correspondence we can build ordered vectors of part descriptions, and conduct the classification in this shared feature space. (Better seen in color)

This discrimination paradigm cannot easily generalize to the classification of basic level objects, mostly because these objects do not share common informative parts, and therefore cannot be efficiently compared using an ordered vector of fixed parts. This problem is partially addressed in a more recent line of work (e.g., [5, 6, 2, 7, 9]), where part-based generative models of objects are learned directly from images. In this paradigm objects are modeled as a set of parts with spatial relations between them. The models are learnt and applied to images, which are represented as unordered feature sets (usually image patches). Learning algorithms developed within this paradigm are typically more complex and less efficient than traditional classifiers learnt in some fixed vector space. However, given the characteristics of human categorization discussed above, this seems to be the correct paradigm to address the classification of basic level categories.

These considerations suggest that sub-ordinate classification should be solved using a two stage method: First we should learn a generative model for the basic category. Using such a model, the object parts should be identified in each image, and their descriptions can be concatenated into an ordered vector. In a second stage, the distinction between subordinate classes can be done by applying standard machine learning tools, like SVM, to the resulting ordered vectors. In this framework, the model learnt in stage 1 is used to solve the correspondence problem: features in the same entry in two different image vectors correspond since they implement the same part. Using this relatively high level representation, the distinction between subordinate categories may be expected to get easier.

Similar notions, of constructing discriminative classifiers on top of generative models, have been recently proposed in the context of object localization [10] and class recognition [7]. The main motivation in these papers was to provide discriminative power to a generative model, optimized by maximum likelihood. Thus the discriminative classifier for a class in [7, 10] uses a generative model of the same class as a representation scheme.[1] In contrast, in this work we use a recent learning algorithm, which already learns a generative relational model of basic categories using a discriminative boosting technique [2]. The new element in our approach is in the learning of a model of one class (the more general basic level category) to allow the efficient discrimination of another class (the more specific sub-ordinates).

Thus our main contribution lies the use of objcet hierarchy, where we represent sub-ordinate classes using models of the more general, basic level class. The approach relies on a specific form of knowledge transfer between classes, and as such it is an instance of the 'learning-to-learn' paradigm. There are several potential benefits to this approach. First and most important is improved accuracy, especially when training data is scarce. For an under-sampled sub-ordinate class, the basic level model can be learnt from a larger sample, leading to a more stable representation for the second stage SVM and lower error rate. A second advantage becomes apparent when scalability is considered: A system which needs to discriminate between many subordinate classes will have to learn and keep considerably less models (only one for each basic level class) if built according to our proposed

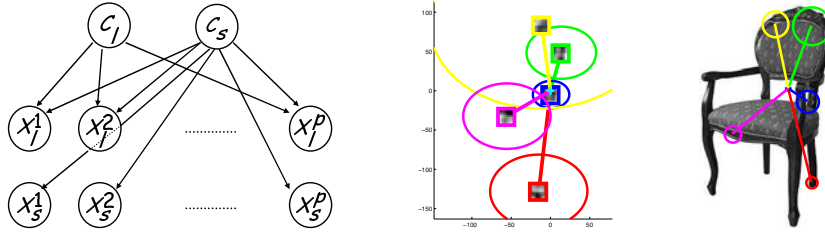

Figure 2: **Left** A Bayesian network specifying the dependencies between the hidden variables $C_l, C_s$ and the parts scale and location $X_l^k, X_s^k$ for $k = 1, .., P$. The part appearance variables $X_a^k$ are independent, and so they do not appear in this network. **Middle** The spatial relations between 5 parts from a learnt chair model. The cyan cross indicates the position of the hidden object center $c_l$. **Right** The implementations of the 5 parts in a chair image. (Better seen in color)

approach. Such a system can better cope with new subordinate classes, since learning to identify a new class may rely on existing basic class models.

Typically the learning of generative models from unsegmented images is exponential in the number of parts and features [5, 6]. This significantly limits the richness of the generative model, to a point where it may not contain enough detail to distinguish between subclass instances. Alternatively, rich models can be learnt from images with part segmentations [4, 9], but obtaining such training data requires a lot of human labor. The algorithm we use in this work, presented in [2], learns from unsegmented images, and its complexity is linear in the number of model parts and image features. We can hence learn models with many parts, providing a rich object description. In section 3 we discuss the importance of this property.

We briefly describe the model learning algorithm in Section 2.1. The details of the two-stage method are then described in Section 2.2. In Section 3 we describe experiments with sub-classes from six basic level categories. We compare our proposed approach, called BLP (Basic Level Primacy), to a one-stage approach. We also compare to another two-stage approach, called SLP (Subordinate Level Primacy), in which discrimination is done based on a model of the subordinate class. In most cases, the results support our claim and demonstrate the superiority of the BLP method.

## 2 Algorithms

To learn class models, we use an efficient learning method briefly reviewed in Section 2.1. Section 2.2 describes the techniques we use for subclass recognition.

### 2.1 Efficient learning of object class models

The learning method from [2] learns a generative relational object model, but the model parameters are discriminatively optimized using an extended boosting process. The class model is learnt from a set of object images and a set of background images. Image $I$ is represented using an unordered feature set $F(I)$ with $N_f$ features extracted by the Kadir & Brady feature detector [8]. The feature set usually contains several hundred features in various scales, with considerable overlap. Features are normalized to uniform size, zero mean and unit variance. They are then represented using their first 15 DCT coefficients, augmented by the image location of the feature and its scale.

The object model is a generative part-based model with $P$ parts (see example in Fig. 2b), where each part is implemented by a single image feature. For each part, its appearance, location and scale are modeled. The appearance of parts is assumed to be independent, while their location and scale are relative to the unknown object location and scale. This dependence is captured by a Bayesian network model, shown in Fig. 2a. It is a star-like model, where the center node is a 3-dimensional hidden node $C = (\vec{C}_l, C_s)$, with the vector $\vec{C}_l$ denoting the unknown object location and the scalar $C_s$ denoting its unknown scale. All the components of the part model, including appearance, relative location and relative log-scale, are modeled using Gaussian distributions with a (scaled) identity covariance matrix.

Based on this model and some simplifying assumptions, the likelihood ratio test classifier is approximated by

$$f(I) = \max_C \sum_{k=1}^{P} \max_{x \in F(I)} \log p(x|C, \theta^k) - \nu \qquad (1)$$

This classifier compares the first term, which represents the approximated image likelihood, to a threshold $\nu$. The likelihood term approximates the image likelihood using the MAP interpretation of the model in the image, i.e., it is determined by the single best implementation of model parts by image features. This MAP solution can be efficiently found using standard message passing in time linear in the number of parts $P$ and the number of image features $N_f$. However, Maximum Likelihood (ML) parameter optimization cannot be used, since the approximation permits part repetition, and as a result the ML solution is vulnerable to repetitive choices of the same part. Instead, the model is optimized to minimize a discriminative loss function.

Specifically, labeling object images by $+1$ and background images by $-1$, the learning algorithm tries to minimize the exp loss of the margin, $L(f) = \sum_{i=1}^{N} \exp(-y_i f(I_i))$, which is the loss minimized by the Adaboost algorithm [12]. The optimization is done using an extended 'relational' boosting scheme, which generalizes the boosting technique to classifiers of the form (1).

In the relational boosting algorithm, the weak hypotheses (summands in Eq. (1)) are not merely functions of the image $I$, but depend also on the hidden variable $C$, which captures the unknown location and scale of the object. In order to find good part hypotheses, the weak learner is given the best current estimate of $C$, and uses it to guide the search for a discriminative part hypothesis. After the new part hypothesis is added to the model, $C$ is re-inferred and the new estimate is used in the next boosting round. Additional tweaks are added to improve class recognition results, including a gradient descent weak learner and a feedback loop between the optimization of the a weak hypothesis and its weight.

## 2.2 Subclass recognition

As stated in the introduction, we approach subclass recognition using a two-stage algorithm. In the first stage a model of the basic level class is applied to the image, and descriptors of the identified parts are concatenated into an ordered vector. In the second stage the subclass label is determined by feeding this vector into a classifier trained to identify the subclass. We next present the implementation details of these two stages.

**Class model learning** Subclass recognition in the proposed framework depends on part consistency across images, and it is more sensitive to part identification failures than the original class recognition task. Producing an informative feature vector is only possible using a rich model with many stable parts. We therefore use a large number of features ($N_f = 400$) per image, and a relatively fine grid of $C$ values, with $10 \times 10$ locations over the entire image and 3 scales (a total of $N_c = 300$ possible values for the hidden variable $C$). We also learn large models with $P = 60$ parts.[2] Note that such large values for $N_f$ and $P$ are not possible in a purely generative framework such as [5, 6] due to the prohibitive computational learning complexity of $O(N_f^P)$.

In [2], model parts are learnt using a gradient based weak learner, which tends to produce exaggerated part location models to enhance its discriminative power. In such cases parts are modeled as being unrealistically far from the object center. Here we restrict the dynamics of the location model in order to produce more realistic and stable parts. In addition, we found out experimentally that when the data contains object images with rich backgrounds, performance of subclass recognition and localization is improved when using models with increased relative location weight. Specifically, a part hypothesis in the model includes appearance, location and scale components with relative weights $\lambda_i/(\lambda_1 + \lambda_2 + \lambda_3)$, $i = 1, 2, 3$, learnt automatically by the algorithm. We multiply $\lambda_2$ of all the parts in the learnt model by a constant factor of 10 when learning from images with rich background. Probabilistically, such increase of $\lambda_2$ amounts to smaller location covariance, and hence to stricter demands on the accuracy of the relative locations of parts.

**Subclass discrimination**    Given a learnt object model and a new image, we match for each model part the corresponding image feature which implements it in the MAP solution. We then build the feature vector, which represents the new image, by concatenating all the feature descriptors implementing parts $1, ..P$. Each feature is described using a 21-dimensional descriptor including:

- The 15 DCT coefficients describing the feature.
- The relative (x,y) location and log-scale of the feature (relative to the computed MAP value of $C$).
- A normalized mean of the feature $(m - \hat{m})/std(m)$ where $m$ is the feature's mean (over feature pixels), and $\hat{m}, std(m)$ are the empirical mean and std of $m$ over the $P$ parts in the image.
- A normalized logarithm of feature variance $(v - \hat{v})/std(v)$ with $v$ the logarithm of the feature's variance (over feature pixels) and $\hat{v}, std(v)$ the empirical mean and std of $v$ over image parts.
- The log-likelihood of the feature (according to the part's model).

In the end, each image is represented by a vector of length $21 \times P$. The training set is then normalized to have unit variance in all the dimensions, and the standard deviations are stored in order to allow for identical scaling of the test data. Vector representations are prepared in this manner for a training sample including objects from the sub-ordinate class, objects from other sub-ordinate classes of the same basic category, and background images. Finally, a linear SVM [3] is trained to discriminate the target subordinate class images from all other images.

## 3  Experimental results

**Methods:**    In our experiments, we regard subclass recognition as a binary classification problem in a retrieval scenario. Specifically, The learning algorithm is given a sample of background images, and a sample of unsegmented class images. Images are labeled by the subclass they represent, or as background if they do not contain any object from the inclusive class. The algorithm is trained to identify a specific subclass. In the test phase, the algorithm is given another sample from the same distribution of images, and is asked to identify images from the specific subclass.

Several methodological problems arise in this scenario. First, subclasses are often not mutually exclusive [13], and in many cases there are borderline instances which are inherently ambiguous. This may lead to an ill-defined classification problem. We avoid this problem in the current study by filtering the data sets, leaving only instances with clear-cut subclass affiliation. The second problem concerns performance measurements. The common measure used in related work is the equal error rate of the ROC curve (denoted here EER), i.e., the error obtained when the rate of false positives and the rate of false negatives are equal. However, as discussed in [1], this measure is not well suited for a detection scenario, where the number of positive examples is much smaller than the number of negative examples. A better measure appears to be the equal error rate of the recall-precision curve (denoted here RPC). Subclass recognition has the same characteristics, and we therefore prefer the RPC measure; for completeness, and since the measures do not give qualitatively different results, the EER score is also provided.

**The algorithms compared:**    We compare the performance of the following three algorithms:

- *Basic Level Primacy (BLP)* - The two-stage method for subclass recognition described above, in which a model of the basic level category is used to form the vector representation.
- *Subordinate level primacy (SLP)* - A two-stage method for subclass recognition, in which a model of the sub-ordinate level category is used to form the vector representation.
- *One stage method* - The classification is based on the likelihood obtained by a model of the sub-ordinate class.

The three algorithms use the same training sample in all the experiments. The class models in all the methods were implemented using the algorithm described in Section 2.1, with exactly the same

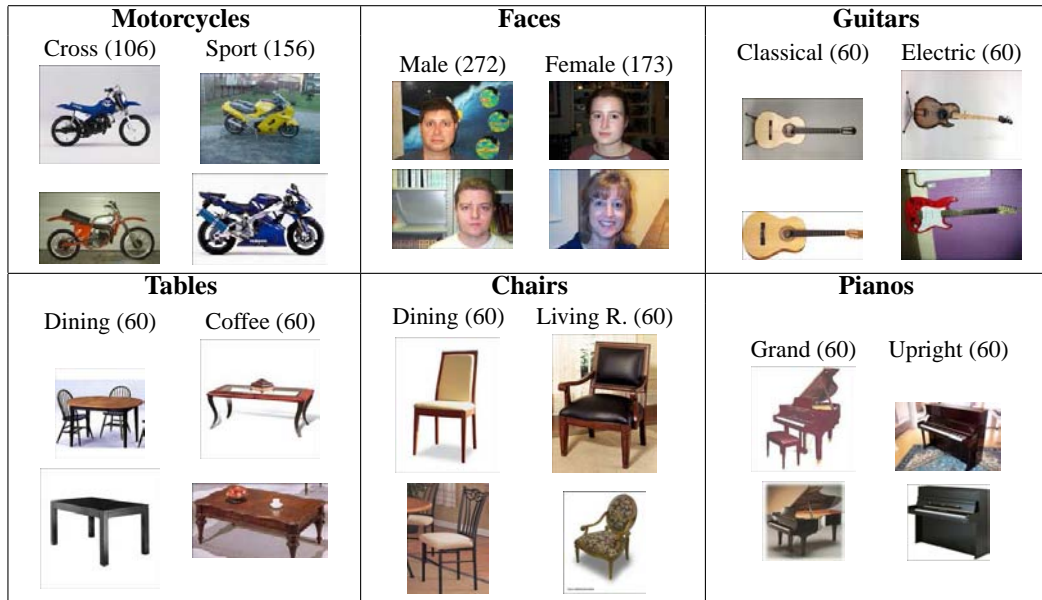

Figure 3: Object images from the subclasses learnt in our experiments. We used 12 subclasses of 6 basic classes. The number of images in each subclass is indicated in the parenthesis next to the subclass name. Individual faces were also considered as subclasses, and the males and females subclasses above include a single example from 4 such individuals.

parameters (reported in section 2.2). This algorithm is competitive with current state-of-the-art methods in object class recognition [2].

The third and the second method learn a different model for each subordinate category, and use images from the other sub-ordinate classes as part of the background class during model learning. The difference is that in the third method, classification is done based on the model score (as in [2]), and in the second the model is only used to build a representation, while classification is done with an SVM (as in [7]). The first and second method both employ the distinction between a representation and classification, but the first uses a model of the basic category, and so tries to take advantage of the structural similarity between different subordinate classes of the same basic category.

**Datasets** We have considered 12 subordinate classes from 6 basic categories. The images were obtained from several sources. Specifically, we have re-labeled subsets of the Caltech Motorcycle and Faces database[3], to obtain the subordinates of sport and cross motorcycles, and male and female faces. For these data sets we have increased the weight of the location model, as mentioned in section 2.2. We took the subordinate classes of grand piano and electric guitar from the Caltech 101 dataset [4] and supplemented them with classes of upright piano and classical guitar collected using google images. Finally, we used subsets of the chairs and furniture background used in [2][5] to define classes of dining and living room chairs, dining and coffee tables. Example images from the data sets can be seen in Fig. 3. In all the experiments, the Caltech office background data was used as the background class. In each experiment half of the data was used for training and the other half for test.

In addition, we have experimented with individual faces from the Caltech faces data set. In this experiment each individual is treated as a sub-ordinate class of the Faces basic class. We filtered the faces data to include only people which have at least 20 images. There were 19 such individuals, and we report the results of these experiments using the mean error.

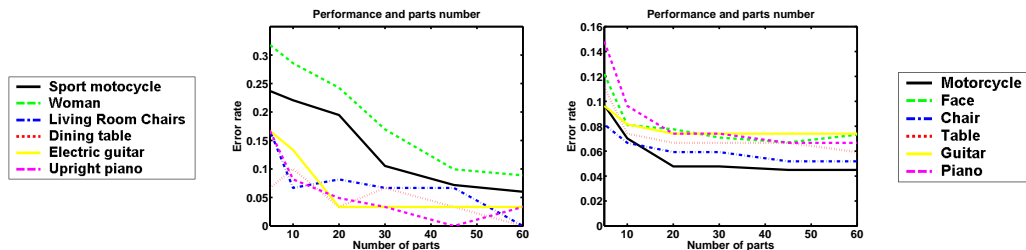

Figure 4: **Left:** RPC error rates as a function of the number of model parts $P$ in the two-stage BLP method, for $5 \leq P \leq 60$. The curves are presented for 6 representative subclasses, one from each basic level category presented in Fig. 3 **Right:** classification error of the first stage classifier as a function of $P$. This graph reports errors for the 6 basic level models used in the experiments reported on the left graph. In general, while adding only a minor improvement to inclusive class recognition, adding parts beyond 30 significantly improves subclass recognition performance.

**Classification results** Table 1 summarizes the classification results. We can see that both two-stage methods perform better than the one-stage method. This shows the advantage of the distinction between representation and classification, which allows the two-stage methods to use the more powerful SVM classifier. When comparing the two two-stage methods, BLP is a clear winner in 7 of the 13 experiments, while SLP has a clear advantage only in a single case. The representation based on the basic level model is hence usually preferable for the fine discriminations required. Overall, the BLP method is clearly superior to the other two methods in most of the experiments, achieving results comparable or superior to the others in 11 of the 13 problems. It is interesting to note that the SLP and BLP show comparable performance when given the individual face subclasses. Notice however, that in this case BLP is far more economical, learning and storing a single face model instead of the 19 individual models used by SLP.

| Subclass | One stage method | | Subordinate level primacy | | Basic level primacy | |
|---|---|---|---|---|---|---|
| Cross motor. | 14.5 | (12.7) | 9.9 | (3.5) | **5.5** | **( 1.7)** |
| Sport motor. | 10.5 | (5.7) | 6.6 | (5.0) | **4.6** | **(2.6)** |
| Males | **20.6** | **(12.4)** | 24.7 | (19.4) | 21.9 | (16.7) |
| Females | 10.6 | (7.1) | 10.6 | (7.9) | **8.2** | **(5.9)** |
| Dining chair | 6.7 | (3.6 ) | **0** | **(0)** | **0** | **(0)** |
| Living room chair | 6.7 | (6.7) | **0** | **(0)** | **0** | **(0)** |
| Coffee table | 13.3 | (6.2) | 8.4 | (6.7) | **3.3** | **(3.6)** |
| Dining table | 6.7 | (3.6) | 4.9 | (3.6) | **0** | **(0)** |
| Classic guitar | 4.9 | (3.1) | **3.3** | **(0.5)** | 6.7 | (3.1) |
| Electric guitar | 6.7 | (3.6) | **3.3** | (3.6) | **3.3** | **(2.6)** |
| Grand piano | 10.0 | **(3.6)** | 10.0 | **(3.6)** | **6.7** | (4.0) |
| Upright piano | **3.3** | (3.6) | 10.0 | (6.7) | **3.3** | **(0.5)** |
| Individuals | 27.5* | (24.8)* | **17.9*** | (7.3)* | 19.2* | **(6.5)*** |

Table 1: Error rates (in percents), when separating subclass images from non-subclass and background images. The main numbers indicate equal error rate of the recall precision curve (RPC). Equal error rate of the ROC (EER) are reported in parentheses. The best result in each row is shown in bold. For the individuals subclasses, the mean over 19 people is reported (marked by ∗). Overall, the BLP method shows a clear advantage.

**Performance as a function of number of parts** Fig. 4 presents errors as a function of $P$, the number of class model parts. The graph on the left plots RPC errors of the two stage BLP method on 6 representative data sets. The graph on the right describes the errors of the first stage class models in the task of discriminating the basic level classes background images. While the performance of inclusive class recognition stabilizes after $\sim 30$ parts, the error rates in subclass recognition continue to drop significantly for most subclasses well beyond 30 parts. It seems that while later boosting rounds have minor contribution to class recognition in the first stage of the algorithm, the added parts enrich the class representation and allow better subclass recognition in the second stage.

## 4 Summary and Discussion

We have addressed in this paper the challenging problem of distinguishing between subordinate classes of the same basic level category. We showed that two augmentations contribute to performance when solving such problems: First, using a two-stage method where representation and classification are solved separately. Second, using a larger sample from the more general basic level category to build a richer representation. We described a specific two stage method, and experimentally showed its advantage over two alternative variants.

The idea of separating representation from classification in such a way was already discussed in [7]. However, our method is different both in motivation and in some important technical details. Technically speaking, we use an efficient algorithm to learn the generative model, and are therefore able to use a rich representation with dozens of parts (in [7] the representation typically includes 3 parts). Our experiments show that the large number of model parts i a critical for the success of the two stage method.

The more important difference is that we use the hierarchy of natural objects, and learn the representation model for a more general class of objects - the basic level class (BLP). We show experimentally that this is preferable to using a model of the target subordinate (SLP). This distinction and its experimental support is our main contribution. Compared with the more traditional SLP method, the BLP method suggested here enjoys two significant advantages. First and most importantly, its accuracy is usually superior, as demonstrated by our experiments. Second, the computational efficiency of learning is much lower, as multiple SVM training sessions are typically much shorter than multiple applications of relational model learning. In our experiments, learning a generative relational model per class (or subclass) required 12-24 hours, while SVM training was typically done in a few seconds. This advantage is more pronounced as the number of subclasses of the same class increases. As scalability becomes an issue, this advantage becomes more important.

## Footnotes

[1]An exception to this rule is the Caltech 101 experiment of [7], but there the discriminative classifiers for all 101 classes relies on the same two arbitrary class models.

[2]In comparison, class recognition in [2] was done with $N_f = 200$, $N_c = 108$ and $P = 50$.

[3] Available at http://www.robots.ox.ac.uk/ vgg/data.html.

[4] Available at http://www.vision.caltech.edu/feifeili/Datasets.htm

[5] Available at http://www.cs.huji.ac.il/ aharonbh/#Data.

## References

[1] S. Agarwal and D. Roth. Learning a sparse representation for object detection. In *ECCV*, 2002.

[2] A. Bar-Hillel, T. Hertz, and D. Weinshall. Efficient learning of relational object class models. In *ICCV*, 2005.

[3] G.C. Cawley. MATLAB SVM Toolbox [http://theoval.sys.uea.ac.uk/~gcc/svm/toolbox].

[4] P. Feltzenswalb and D. Hutenlocher. Pictorial structures for object recognition. *IJCV*, 61:55–79, 2005.

[5] R. Fergus, P. Perona, and A. Zisserman. Object class recognition by unsupervised scale invariant learning. In *CVPR*, 2003.

[6] R. Fergus, P. Perona, and A. Zisserman. A sparse object category model for efficient learning and exhaustive recognition. In *CVPR*, 2005.

[7] AD. Holub, M. Welling, and P. Perona. Combining generative models and fisher kernels for object class recognition. In *International Conference on Computer Vision (ICCV)*, 2005.

[8] T. Kadir and M. Brady. Scale, saliency and image description. *IJCV*, 45(2):83–105, November 2001.

[9] B. Leibe, A. Leonardis, and B. Schiele. Combined object categorization and segmentation with an implicit shape model. In *ECCV workshop on statistical learning in computer vision*, 2004.

[10] Fritz M., Leibe B., Caputo B., and Schiele B. Integrating representative and discriminative models for object category detection. In *ICCV*, pages 1363–1370, 2005.

[11] E. Rosch, C.B. Mervis, W.D. Gray, D.M. Johnson, and P. Boyes-Braem. Basic objects in natural categories. *Cognitive Psychology*, 8:382–439, 1976.

[12] R.E. Schapire and Y. Singer. Improved boosting using confidence-rated predictions. *Machine Learning*, 37(3):297–336, 1999.

[13] B. Tversky and K. Hemenway. Objects, parts, and categories. *Journal of Experimental Psychology: General*, 113(2):169–197, 1984.
